# Incorporating Contextual Information in White Blood Cell Identification

**Xubo Song\***
Department of Electrical Engineering
California Institute of Technology
Pasadena, CA 91125
xubosong@fire.work.caltech.edu

**Yaser Abu-Mostafa**
Dept. of Electrical Engineering
and Dept. of Computer Science
California Institute of Technology
Pasadena, CA 91125
Yaser@over.work.caltech.edu

**Joseph Sill**
Computation and Neural Systems Program
California Institute of Technology
Pasadena, CA 91125
joe@busy.work.caltech.edu

**Harvey Kasdan**
International Remote Imaging Systems
9162 Eton Ave.,
Chatsworth, CA 91311

## Abstract

In this paper we propose a technique to incorporate contextual information into object classification. In the real world there are cases where the identity of an object is ambiguous due to the noise in the measurements based on which the classification should be made. It is helpful to reduce the ambiguity by utilizing extra information referred to as context, which in our case is the identities of the accompanying objects. This technique is applied to white blood cell classification. Comparisons are made against "no context" approach, which demonstrates the superior classification performance achieved by using context. In our particular application, it significantly reduces false alarm rate and thus greatly reduces the cost due to expensive clinical tests.

# 1   INTRODUCTION

One of the most common assumptions made in the study of machine learning is that the examples are drawn *independently* from some joint input-output distribution. There are cases, however, where this assumption is not valid. One application where the independence assumption does not hold is the identification of white blood cell images. Abnormal cells are much more likely to appear in bunches than in isolation. Specifically, in a sample of several hundred cells, it is more likely to find either no abnormal cells or many abnormal cells than it is to find just a few.

In this paper, we present a framework for pattern classification in situations where the independence assumption is not satisfied. In our case, the identity of an object is dependent of the identities of the accompanying objects, which provides the contextual information. Our method takes into consideration the joint distribution of all the classes, and uses it to adjust the object-by-object classification.

In section 2, the framework for incorporating contextual information is presented, and an efficient algorithm is developed. In section 3 we discuss the application area of white blood cell classification, and address the importance of using context for this application. Empirical testing results are shown in Section 4, followed by conclusions in Section 5.

# 2   INCORPORATING CONTEXTUAL INFORMATION INTO CLASSIFICATION

## 2.1   THE FRAMEWORK

Let $x_i$ be the feature vector of an object , and $c_i = c(x_i)$ be the classification for $x_i, i = 1, ... N$, where N is the total number of objects. $c_i \in \{1, ..., D\}$, where $D$ is the number of total classes.

According to Bayes rule,

$$p(c|x) = \frac{p(x|c)p(c)}{p(x)}$$

It follows that the "with context" *a posteriori* probability of the class labels of all the objects assuming values $c_1, c_2, ..., c_N$, given all the feature vectors, is

$$p(c_1, c_2, ..., c_N | x_1, x_2, ..., x_N) = \frac{p(x_1, x_2, ..., x_N | c_1, c_2, ..., c_N)p(c_1, c_2, ..., c_N)}{p(x_1, x_2, ..., x_N)} \quad (1)$$

It is reasonable to assume that the feature distribution given a class is independent of the feature distributions of other classes, *i.e.*,

$$p(x_1, x_2, ..., x_N | c_1, c_2, ..., c_N) = p(x_1|c_1)...p(x_N|c_N)$$

Then Equation (1) can be rewritten as

$$p(c_1, c_2, ..., c_N | x_1, x_2, ..., x_N) = \frac{p(x_1|c_1)...p(x_N|c_N)p(c_1, c_2, ..., c_N)}{p(x_1, x_2, ..., x_N)} \quad (2)$$

$$= \frac{p(c_1|x_1)...p(c_N|x_N)p(x_1)...p(x_N)p(c_1, c_2, ..., c_N)}{p(c_1)...p(c_N)p(x_1, x_2, ..., x_N)}$$

where $p(c_i|\mathbf{x}_i)$ is the "no context" object-by-object Bayesian *a posteriori* probability, and $p(c_i)$ is the *a priori* probability of the classes, $p(\mathbf{x}_i)$ is the marginal probability of the features, and $p(\mathbf{x}_1, \mathbf{x}_2, ..., \mathbf{x}_N)$ is the joint distribution of all the feature vectors.

Since the features $(\mathbf{x}_1, \mathbf{x}_2, ..., \mathbf{x}_N)$ are given, $p(\mathbf{x}_1, \mathbf{x}_2, ..., \mathbf{x}_N)$ and $p(\mathbf{x}_i)$ are constant,

$$p(c_1, ..., c_N | \mathbf{x}_1, ..., \mathbf{x}_N) \propto p(c_1|\mathbf{x}_1)...p(c_N|\mathbf{x}_N)\frac{p(c_1, ..., c_N)}{p(c_1)...p(c_N)}$$

$$= p(c_1|\mathbf{x}_1)...p(c_N|\mathbf{x}_N)\rho(c_1, c_2, ..., c_N)$$

where

$$\rho(c_1, c_2, ..., c_N) \equiv \frac{p(c_1, c_2, ..., c_N)}{p(c_1)...p(c_N)} \tag{3}$$

The quantity $\rho(c_1, c_2, ..., c_N)$, which we call *context ratio* and through which the context plays its role, captures the dependence among the objects. In the case where all the objects are independent, $\rho(c_1, c_2, ..., c_N)$ equals one – there will be no context. In the dependent case, $\rho(c_1, c_2, ..., c_N)$ will not equal one, and the context has an effect on the classifications.

We deal with the application of object classification where it is the count in each class, rather than the particular ordering or numbering of the objects, that matters. As a result, $\rho(c_1, c_2, ..., c_N)$ is only a function of the count in each class. Let $N_d$ be the count in class $d$, and $\nu_d = \frac{N_d}{N}$, $d = 1..., D$,

$$\rho(c_1, c_2, ..., c_N) = \frac{p(c_1, c_2, ..., c_N)}{p(c_1)...p(c_N)}$$

$$= \frac{N_1!...N_D! \; p(\nu_1, \nu_2, ..., \nu_D)}{N! \; P_1^{N\nu_1}...P_D^{N\nu_D}} \tag{4}$$

$$= \rho(\nu_1, ..., \nu_D)$$

where $P_d$ is the prior distribution of class $d$, for $d = 1, ...D$. $\sum_{d=1}^{D} N_d = N$ and $\sum_{d=1}^{D} \nu_d = 1$.

The decision rule is to choose class labels $\hat{c}_1, \hat{c}_2, ..., \hat{c}_N$ such that

$$(\hat{c}_1, \hat{c}_2, ..., \hat{c}_N) = \operatorname*{argmax}_{(c_1, c_2, ..., c_N)} \; p(c_1, c_2, ..., c_N | \mathbf{x}_1, \mathbf{x}_2, ..., \mathbf{x}_N) \tag{5}$$

When implementing the decision rule, we need to compute and compare $D^N$ cases for Equation 5. In the case of white blood cell recognition, $D = 14$ and $N$ is typically around 600, which makes it virtually impossible to implement.

In many cases, additional constraints can be used to reduce computation, as is the case in white blood cell identification, which will be demonstrated in the following section.

## 3  WHITE BLOOD CELL RECOGNITION

Leukocyte analysis is one of the major routine laboratory examinations. The utility of leukocyte classification in clinical diagnosis relates to the fact that in various physiological and pathological conditions the relative percentage composition of the blood leukocytes

changes. An estimate of the percentage of each class present in a blood sample conveys information which is pertinent to the hematological diagnosis. Typical commercial differential WBC counting systems are designed to identify five major mature cell types. But blood samples may also contain other types of cells, *i.e.* immature cells. These cells occur infrequently in normal specimen, and most commercial systems will simply indicate the presence of these cells because they can't be individually identified by the systems. But it is precisely these cell types that relate to the production rate and maturation of new cells and thus are important indicators of hematological disorders. Our system is designed to differentiate fourteen WBC types which includes the five major mature types: segmented neutrophils, lymphocytes, monocytes, eosinophils, and basophils; and the immature types: bands (unsegmented neutrophils), metamyelocytes, myelocytes, promyelocytes, blasts, and variant lymphocytes; as well as nucleated red blood cells and artifacts. Differential counts are made based on the cell classifications, which further leads to diagnosis or prognosis.

The data was provided by IRIS, Inc. Blood specimens are collected at Harbor UCLA Medical Center from local patients, then dyed with Basic Orange 21 metachromatic dye supravital stain. The specimen is then passed through a flow microscopic imaging and image processing instrument, where the blood cell images are captured. Each image contains a single cell with full color. There are typically 600 images from each specimen. The task of the cell recognition system is to categorize the cells based on the images.

## 3.1 PREPROCESSING AND FEATURE EXTRACTION

The size of cell images are automatically tailored according to the size of the cell in the images. Images containing larger cells have bigger sizes than those with small cells. The range varies from 20x20 to 40x40 pixels. The average size is around 25x25. See Figure 3.1. At the preprocessing stage, the images are segmented to set the cell interior apart from the background. Features based on the interior of the cells are extracted from the images. The features include size, shape, color [1] and texture. See Table 1 for the list of features. [2]

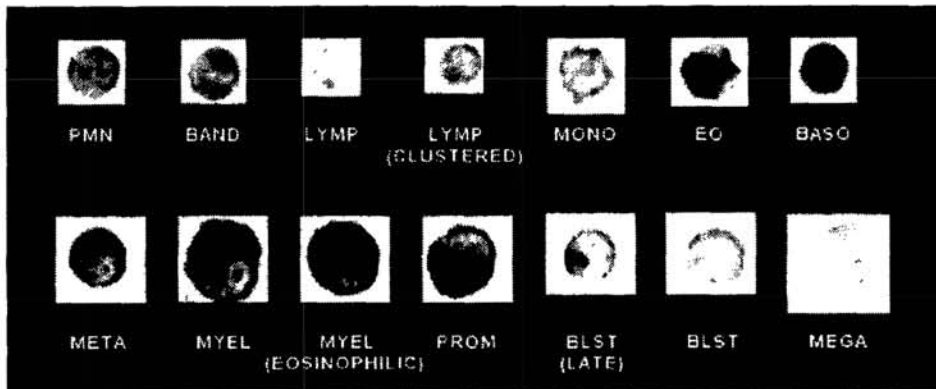

Figure 1: Example of some of the cell images.

## 3.2 CELL-BY-CELL CLASSIFICATION

The features are fed into a nonlinear feed-forward neural network with 20 inputs, 15 hidden units with sigmoid transfer functions, and 14 sigmoid output units. A cross-entropy error

| feature number | feature description |
|---|---|
| 1 | cell area |
| 2 | number of pixels on cell edge |
| 3 | the 4th quantile of red-blue distribution |
| 4 | the 4th quantile of green-red distribution |
| 5 | the median of red-blue distribution |
| 6 | the median of green-red distribution |
| 7 | the median of blue-green distribution |
| 8 | the standard deviation of red-blue distribution |
| 9 | the standard deviation of green-red distribution |
| 10 | the standard deviation of blue-green distribution |
| 11 | the 4th quantile of red distribution |
| 12 | the 4th quantile of green distribution |
| 13 | the 4th quantile of blue distribution |
| 14 | the median of red distribution |
| 15 | the median of green distribution |
| 16 | the median of blue distribution |
| 17 | the standard deviation of red distribution |
| 18 | the standard deviation of green distribution |
| 19 | the standard deviation of blue distribution |
| 20 | the standard deviation of the distance from the edge to the mass center |

function is used in order to give the output a probability interpretation. Denote the input feature vector as $\mathbf{x}$, the network outputs a $D$ dimensional vector ( $D = 14$ in our case) $\mathbf{p} = \{p(d|\mathbf{x})\}, d = 1, ..., D$, where $p(d|\mathbf{x})$ is

$$p(d|\mathbf{x}) = Prob(\text{ a cell belongs to class } d| \text{ feature } \mathbf{x})$$

The decision made at this stage is

$$d(\mathbf{x}) = \underset{d}{\operatorname{argmax}} \ p(d|\mathbf{x})$$

### 3.3 COMBINING CONTEXTUAL INFORMATION

The "no-context" cell-by-cell decision is only based on the features presented by a cell, without looking at any other cells. When human experts make decisions, they always look at the whole specimen, taking into consideration the identities of other cells and adjusting the cell-by-cell decision on a single cell according to the company it keeps. On top of the visual perception of the cell patterns, such as shape, color, size, texture, etc., comparisons and associations, either mental or visual, with other cells in the same specimen are made to infer the final decision. A cell is assigned a certain identity if the company it keeps supports that identity. For instance, the difference between lymphocyte and blast can be very subtle sometimes, especially when the cell is large. A large unusual mononuclear cell with the characteristics of both blast and lymphocyte is more likely to be a blast if surrounded by or accompanied by other abnormal cells or abnormal distribution of the cells.

This scenario fits in the framework we described in section 2. The Combining Contextual Information algorithm was used as the post-precessing of the cell-by-cell decisions.

## 3.4 OBSERVATIONS AND SIMPLIFICATIONS

Direct implementation of the proposed algorithm is difficult due to the computational complexity. In the application of WBC identification, simplification is possible. We observed the following: First, we are primarily concerned with one class blast, the presence of which has clinical significance. Secondly, we only confuse blast with another class lymphocyte. In other words, for a potential blast, $p(\text{blast}|\mathbf{x}) \gg 0$, $p(\text{lymphocyte}|\mathbf{x}) \gg 0$, $p(\text{any other class}|\mathbf{x}) \approx 0$. Finally, we are fairly certain about the classification of all other classes, *i.e.* $p(\text{a certain class}|\mathbf{x}) \approx 1$, $p(\text{any other class}|\mathbf{x}) \approx 0$. Based on the above observations, we can simplify the algorithm, instead of doing an exhaustive search.

Let $p_i^d = p(c_i = d|\mathbf{x}_i), i = 1, ..., N$. More specifically, let $p_i^B = p(\text{blast}|\mathbf{x}_i)$, $p_i^L = p(\text{lymphocyte}|\mathbf{x}_i)$ and $p_i^* = p(\text{class} * |\mathbf{x}_i)$ where $*$ is neither a blast nor a lymphocyte.

Suppose there are $K$ potential blasts. Order the $p_1^B, p_2^B, ..., p_K^B$'s in a descending manner over $i$, such that

$$p_1^B \geq p_2^B \geq ... \geq p_K^B$$

then the probability that there are $k$ blasts is

$$P_B(k) = p_1^B ... p_k^B p_{k+1}^L ... p_K^L \, p_{K+1}^* ... p_N^* \, \rho(\nu_B = \tfrac{k}{N}, \nu_L = \nu_L' + \tfrac{K-k}{N}, \nu_3, ..., \nu_D)$$

where $\nu_L'$ is the proportion of unambiguous lymphocytes and $\nu_3, ..., \nu_D$ are the proportions of the other cell types.

We can compute the $P_B(k)$'s recursively.

$$P_B(0) = p_1^L ... p_K^L \, p_{K+1}^* ... p_N^* \, \rho(\nu_B = 0, \nu_L = \nu_L' + \tfrac{K}{N}, \nu_3, ..., \nu_D)$$

$$P_B(k+1) = P_B(k) \frac{p_{k+1}^B \, \rho(\nu_B = \tfrac{k+1}{N}, \nu_L = \nu_L' + \tfrac{K-k-1}{N}, \nu_3, ..., \nu_D)}{p_{k+1}^L \, \rho(\nu_B = \tfrac{k}{N}, \nu_L = \nu_L' + \tfrac{K-k}{N}, \nu_3, ..., \nu_D)}$$

for k = 1, ..., K-1, and

$$P_B(K) = p_1^B ... p_K^B \, p_{K+1}^* ... p_N^* \, \rho(\nu_B = \tfrac{K}{N}, \nu_L = \nu_L', \nu_3, ..., \nu_D)$$

This way we only need to compute $K$ terms to get $P_B(k)$'s. Pick the optimal number of blasts $k^*$ that maximizes $P_B(k), k = 1, ..., K$.

An important step is to calculate $\rho(\nu_1, ..., \nu_D)$ which can be estimated from the database.

## 3.5 THE ALGORITHM

**Step 1** Estimate $\rho(\nu_1, ..., \nu_D)$ from the database, for $d = 1, ..., D$.

**Step 2** Compute the object-by-object "no context" *a posteriori* probability $p(c_i|\mathbf{x}_i), i = 1, ..., N$, and $c_i \in \{1, ..., D\}$.

**Step 3** Compute $P_B(k)$ and find $k^*$ for $k = 1, ..., K$, and relabel the cells accordingly.

## 4  EMPIRICAL TESTING

The algorithm has been intensively tested at IRIS, Inc. on the specimens obtained at Harbor UCLA medical center. We compared the performances with or without using contextual information on blood samples from 220 specimens (consisting of 13,200 cells). In about 50% of the cases, a false alarm would have occurred had context not been used. Most cells are correctly classified, but a few are incorrectly labelled as immature cells, which raises a flag for the doctors. Change of the classification of the specimen to abnormal requires expert intervention before the false alarm is eliminated, and it may cause unnecessary worry. When context is applied, the false alarms for most of the specimens were eliminated, and no false negative was introduced.

| methods | cell classification | normality identification | false positive | false negative |
|---|---|---|---|---|
| no context | 88% | $\sim 50\%$ | $\sim 50\%$ | 0% |
| with context | 89% | $\sim 90\%$ | $\sim 10\%$ | 0% |

Table 2: Comparison of with and without using contextual information

## 5  CONCLUSIONS

In this paper we presented a novel framework for incorporating contextual information into object identification, developed an algorithm to implement it efficiently, and applied it to white blood cell recognition. Empirical tests showed that the "with context" approach is significantly superior than the "no context" approach. The technique described could be generalized to a number of domains where contextual information plays an essential role, such a speech recognition, character recognition and other medical diagnosis regimes.

**Acknowledgments**

The authors would like to thank the members of Learning Systems Group at Caltech for helpful suggestions and advice: Dr. Amir Atiya, Zehra Cataltepe, Malik Magdon-Ismail, and Alexander Nicholson.

**References**

Richard, M.D., & Lippmann, R.P., (1991) Neural network classifiers estimate Bayesian *a posteriori* probabilities. *Neural Computation* **3**. pp.461-483. Cambridge, MA: MIT Press.

Kasdan, H.K., Pelmulder, J.P., Spolter, L., Levitt, G.B., Lincir, M.R., Coward, G.N., Haiby, S. I., Lives, J., Sun, N.C.J., & Deindoerfer, F.H., (1994) The $WhiteIRIS^{TM}$ Leukocyte differential analyzer for rapid high-precision differentials based on images of cytoprobe-reacted cells. *Clinical Chemistry*. Vol. 40, No. 9, pp.1850-1861.

Haralick, R.M., & Shapiro, L.G.,(1992),*Computer and Robot Vision*, Vol.1, Addison-Welsley.

Aus, H. A., Harms, H., ter Meulen, V., & Gunzer, U. (1987) Statistical evaluation of computer extracted blood cell features for screening population to detect leukemias. In Pierre A. Devijver and Josef Kittler (eds.) *Pattern Recognition Theory and Applications*, pp. 509-518. Springer-Verlag.

Kittler, J., (1987) Relaxation labelling. In Pierre A. Devijver and Josef Kittler (eds.) *Pattern Recognition Theory and Applications*, pp. 99-108. Springer-Verlag.

## Footnotes

\*Author for correspondence.

[1]A color image is decomposed into three intensity images – red, green and blue respectively

[2]The red-blue distribution is the pixel-by-pixel log(red)- log(blue) distribution for pixels in cell interior. The red distribution is the distribution of the red intensity in cell interior.
